# A Benchmark Dataset for Event-Guided Human Pose Estimation and Tracking in Extreme Conditions

**Hoonhee Cho**[*]
KAIST
gnsgnsgml@kaist.ac.kr

**Taewoo Kim**[*]
KAIST
intelpro@kaist.ac.kr

**Yuhwan Jeong**
KAIST
jeongyh98@kaist.ac.kr

**Kuk-Jin Yoon**
KAIST
kjyoon@kaist.ac.kr

## Abstract

Multi-person pose estimation and tracking have been actively researched by the computer vision community due to their practical applicability. However, existing human pose estimation and tracking datasets have only been successful in typical scenarios, such as those without motion blur or with well-lit conditions. These RGB-based datasets are limited to learning under extreme motion blur situations or poor lighting conditions, making them inherently vulnerable to such scenarios. As a promising solution, bio-inspired event cameras exhibit robustness in extreme scenarios due to their high dynamic range and micro-second level temporal resolution. Therefore, in this paper, we introduce a new hybrid dataset encompassing both RGB and event data for human pose estimation and tracking in two extreme scenarios: low-light and motion blur environments. The proposed Event-guided Human Pose Estimation and Tracking in eXtreme Conditions (EHPT-XC) dataset covers cases of motion blur caused by dynamic objects and low-light conditions individually as well as both simultaneously. With EHPT-XC, we aim to inspire researchers to tackle pose estimation and tracking in extreme conditions by leveraging the advantageous of the event camera. Project pages are available at https://github.com/Chohoonhee/EHPT-XC.

## 1 Introduction

Human pose estimation and tracking involve the identification and monitoring of human body parts or significant joints. This task holds paramount importance in comprehending human activities and analyzing movements across various domains, including rehabilitation, sports, augmented/virtual reality, autonomous driving. Consequently, numerous datasets [10, 33, 11, 20, 30] have been dedicated to studying human pose estimation and tracking for various applications. However, despite the dynamic nature of human activity, most datasets assume that the subjects are well-groomed in terms of motion and lighting conditions. Considering the reality of perceiving human motion, most individuals move in a dynamic fashion. Moreover, human activity occurs across various times of the day, exposing individuals to diverse lighting environments. This diversity is directly reflected in the cameras acquiring the data. Ultimately, AI models should strive to predict human body movements even in such varied environments.

To address this, we acquired human pose estimation and tracking dataset that tackle two extreme cases that can occasionally arise when capturing images with cameras. Firstly, we obtained data

---

[*]Equal contribution.

including motion blur that may occur due to the movement of the subject or camera during the exposure time. While videos from professional sports, captured by experts using high-end equipment like gimbals, tend to be clean, videos taken by average users often exhibit blur caused by moving subjects. Additionally, when the camera attempts to track human objects, motion blur caused by the camera may also occur. Therefore, solving human pose estimation and tracking within motion blur is crucial. Secondly, we acquired data captured in low-light environments to perform human pose estimation and tracking under poor lighting conditions. The ability to analyze human actions at low-light condition has been of continuous interest [22, 41, 17], prompting us to capture data in conditions where very little light is present, allowing for scenarios people are barely visible. These two extreme cases are closely related, as increasing the exposure time of the camera to compensate for the low intensity in low-light environments can exacerbate motion blur. Addressing both cases simultaneously is highly practical and reasonable, as they often occur together and solving them together offers a comprehensive solution.

Performing accurate human pose estimation and tracking in these extreme conditions poses a significant challenge. Especially when both situations are present, relying solely on standard cameras may not provide sufficient information. Therefore, we augmented standard cameras with a auxiliary sensor called an event camera [12, 44], also known as a neuromorphic camera. Event cameras mimic the human eye by providing pixel-wise changes asynchronously in an on/off manner. These cameras possess high dynamic range and high temporal resolution, ensuring sufficient data even in low-light conditions and being immune to motion blur. Therefore, to tackle extreme conditions, we introduce the Event-guided Human Pose Estimation and Tracking in eXtreme Conditions (EHPT-XC) dataset and benchmark. To build the dataset, we constructed a multi-camera system to acquire high-resolution RGB-Event paired data, enabling us to freely adjust camera settings for extreme conditions. Additionally, we captured diverse motions and scenarios through experimental participants and manually labeled the acquired data accordingly. The contributions and unique aspects of EHPT-XC datasets are as follows:

- **Multi-human Pose Dataset with Neuromorphic Cameras.** While existing datasets utilizing RGB frames can be collected from various sources such as publicly available human-centric datasets and data shared on platforms like YouTube and the web, data collection using event cameras presents a challenge, particularly for human-centric datasets, as there are no readily available sources. We gathered human data directly using a multi-camera system, making EHPT-XC the *first multi-human pose dataset utilizing neuromorphic cameras*. Moreover, EHPT-XC provides track IDs, enabling its utilization for multi-object tracking. While datasets for event-based single-object tracking [48, 38] exist, EHPT-XC stands as the *pioneering dataset for multi-object tracking using event cameras*.
- **Real-captured Data in Extreme Conditions.** The EHPT-XC dataset aims to address scenarios characterized by low-light conditions and significant motion blur. The absence of multi-human pose estimation and tracking datasets in such conditions stems from the difficulty in directly acquiring data and annotating it due to the challenges posed by degradation conditions. To tackle this, we developed a multi-camera system consisting of a triplet camera configuration, where two cameras capture RGB frames and the remaining one comprises an event camera. One of the RGB cameras was configured to acquire data in low-light environments and/or motion blur by adjusting its settings, while the other was set to capture sharp image under normal lighting conditions.
- **Indoor/outdoor Environments and Various Scenarios.** The EHPT-XC dataset comprises data not only in general scenes but also in dynamic scenarios such as sports, encompassing a variety of indoor and outdoor environments. This further emphasizes the motivation behind these low-light and motion blur conditions. Additionally, we varied the number of people appearing in each sequence to enhance the versatility of the dataset.

## 2 Related Works

### 2.1 Human Pose Estimation and Tracking Datasets in Low-light Conditions.

Several datasets [45, 23, 21, 40] have been proposed to facilitate perception in low-light environments. However, datasets related to human subjects are challenging to acquire and are not actively publicized due to privacy issues. As shown in Table 1, there are only a few datasets related to human pose estimation or tracking in low-light conditions, including ExLPose [17], WIHPD [41], and M$^3$FD [22]. Acquiring data in such extreme conditions exacerbates the difficulty of the annotation process, resulting in an insufficient number of total images and annotations. For example, the MPII

Table 1: Comparison with different datasets for multi-person pose estimation and multi-object tracking. For each dataset we report the number of poses, boxes, as well as the availability of tracking information, people per frame (ppF) for poses, scene type, modalities, and resolution. $unk$ denotes 'unknown'. We categorize the data based on the presence of extreme conditions such as low light and motion blur. △ signifies that something exists, albeit to a small extent, but its intensity is not strong.

| Dataset | Total images | # Scenes | # Poses | # Boxes | Track ids | ppF poses | Indoor + Outdoor | Modality | Resolution | Extreme Conditions Low Light | Motion Blur |
|---|---|---|---|---|---|---|---|---|---|---|---|
| MPII [3] | 25K | 491 | 40K | ✗ | ✗ | 1-17 | ✓ | RGB | 1280×720 | ✗ | ✗ |
| Penn Action [50] | 160K | 2326 | 160K | 160K | ✗ | 1 | ✓ | RGB | 640×480 | ✗ | △ |
| COCO [20] | 200K | 200K | 250K | 500K | ✗ | 1-20 | ✓ | RGB | 640×480 | ✗ | ✗ |
| MOT20 [9] | 13K | 8 | ✗ | 1.65M | ✗ | ✗ | ✓ | RGB | 1920×1080 | ✗ | ✗ |
| PoseTrack21 [10] | 66K | 514 | 177K | 429K | ✓ | 1-13 | ✓ | RGB | 1280×720 | ✗ | △ |
| ExLPose [17] | 3K | 251 | 15K | ✗ | ✗ | 1-26 | ✓ | RGB | 1920×1200 | ✓ | ✗ |
| mRI [2] | 160K | 300 | 160K | 160K | ✗ | 1 | ✗ | RGB+depth+mmWave+IMU | 512×424 | ✗ | ✗ |
| GoPose [26] | 676k | unk | 676k | ✗ | ✗ | 1 | ✗ | RGB+WiFi | 1920×1080 | ✗ | ✗ |
| MM-Fi [43] | 320K | 1080 | 320K | ✗ | ✗ | 1 | ✗ | RGB+depth+LiDAR+mmWave+WiFi | 1280×720 | ✗ | ✗ |
| RELI11D [42] | 239K | 48 | 239K | ✗ | ✗ | 1 | ✓ | RGB+IMU+LiDAR+Event | 1280×800 | ✗ | ✗ |
| JRDB-Pose [33] | 28K | 54 | 636K | 2.8M | ✓ | 1-36 | ✓ | RGB+LiDAR | 752×480 | ✗ | ✗ |
| NTU RGB+D [28] | 57K | 17 | 57K | ✗ | ✗ | 1 | ✗ | RGB+Depth | 512×424 | ✗ | ✗ |
| M³FD [22] | 4K | 8 | ✗ | 34K | ✗ | ✗ | ✗ | RGB+IR | 1024×768 | ✓ | ✗ |
| WIHPD [41] | 2K | $unk$ | 7.3K | ✗ | ✗ | 1-12 | ✓ | RGB+IR | 1280×720 | ✓ | ✗ |
| **EHPT-XC (Ours)** | 16K | 158 | 38K | 38K | ✓ | 1-13 | ✓ | RGB+Event | 1373×928 | ✓ | ✓ |

dataset [3], which was captured only in normal lighting conditions, contains 25K total images and 40K pose annotations. However, datasets like ExLPose and WIHPD, which include captures in low-light environments, have significantly fewer total images, with 3K and 2K, respectively, and pose annotations of 15K and 7.3K, respectively. On the other hand, the proposed EHPT-XC dataset captures diverse scenes despite the extreme conditions, comprising 16K total images and 38K pose annotations, making it comparable to other datasets. Additionally, we captured a dataset that includes images with both reduced visibility, commonly observed in low-light environments, and motion blur, distinguishing it from the ExLPose and WIHPD datasets. The inclusion of event data further distinguishes it, providing richer information for human action understanding.

## 2.2 Human Pose Estimation and Tracking Datasets in Motion Blur.

The dynamic nature of human motion naturally introduces blur into acquired data, which differs in distribution from clean and sharp images. Failure to account for this during the training process can significantly degrade performance during inference. Existing works [53, 24, 25] have also attempted human pose estimation on blurred images based on similar motivations. However, these works have not used real-captured blurred images; instead, they synthesized blur by interpolating high frame rate videos for training and evaluation due to annotation labels. As mentioned in recent deblurring benchmarks [54, 27, 55, 16, 15], this synthesized blur differs from the actual process of blur generation, leading to discrepancies in properties such as continuity and saturation. One alternative approach is to acquire data during exposure time to capture real blur through motion and perform annotation on these blurred frames. However, annotating accurate poses on images degraded by such blur is extremely challenging and, in severe cases, may be impossible. To accurately annotate poses on images generated through real blur processes, we constructed a multi-camera system with synchronization, enabling precise pose annotation even in situations with intense blur. Our EHPT-XC dataset is the first study to enable annotation on the real blur images corresponding to the sharp ground truth, providing accurate human pose and track IDs even in blurred situations.

## 2.3 Event-based Multi-human Pose Estimation and Tracking Dataset.

Recently, various multi-sensor datasets [4, 18, 14, 1, 8, 52] have emerged for human pose estimation across different edge cases. The EHPT-XC dataset, proposed for multi-human pose estimation and tracking using event cameras, stands as the first of its kind. While there are no directly comparable event-based datasets to our work, existing datasets with some degree of similarity include single human pose estimation, such as [57, 5, 39], and single object tracking, such as [38, 31, 35, 47]. These works primarily leverage the high temporal resolution property of the event modality to effectively address tasks. This underscores the value of our multi-modal dataset and highlights our unique challenge in venturing into the first work on multi-human pose estimation and tracking. Particularly, benchmarking low-light and motion blur as extreme conditions effectively utilizing the event modality sets our work apart.

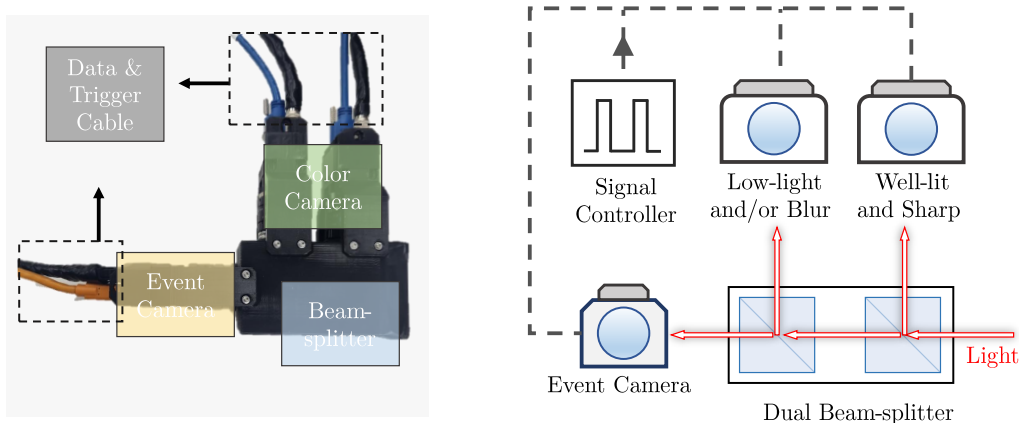

Figure 1: Our triplet camera system comprises two RGB cameras and one event camera, all aligned using two beam splitters. One RGB camera is set to capture sharp frames with short exposure time in a well-lit environment exclusively for annotation purposes, while the other RGB camera is configured to capture low-light images through aperture adjustments or the use of ND filters or extend exposure time for acquiring blurred frames. The event camera was adjusted to ensure the exact same amount of light enters as with the low-light camera by applying the same ND filter and aperture settings.

## 3 Event-guided Human Pose Estimation and Tracking in eXtreme Conditions (EHPT-XC) dataset

EHPT-XC encompasses RGB video frames from 158 diverse sequences, along with pixel-wise aligned and temporally synchronized event streams, and annotations containing 38K 2D keypoints and bounding boxes with track IDs. To ensure accurate annotation in extreme low-light and motion blur environments, we designed a triplet camera system. This system enabled the simultaneous acquisition of RGB frames degraded by low-light and/or blur alongside sharp RGB frames.

### 3.1 Triplet Camera System

To address the difficulty in marking accurate keypoints of human joints in images with strong motion blur, where object boundaries and structures are hard to distinguish, as well as the challenge of annotating in low-light environments where people are barely visible, we devise an approach using an additional RGB camera serving as a reference for annotation alongside RGB and event pair cameras. This additional RGB camera captures sharp and well-lit images, unlike the one recording blurred and/or low-light images, and is used solely for annotation process.

**Camera system with beam splitters.** One of the issues when using multiple cameras to capture the same scene is that each camera has different camera coordinates and image planes, making pixel-wise alignment challenging. To address this, we utilize an optical device called a beam splitter to divide incoming light into two identical beams, allowing two cameras to capture the same scene. Specifically, as shown in Fig. 1, two RGB cameras (BFS-U3-16S2C-CS) and one event camera (Prophesee EVK4) are aligned two 50/50 mirror beam splitters, resulting in a minimal baseline. Our camera system aligns the axes of three cameras but there is still geometric misalignment due to the remaining baseline. To deal with this, we correct the residual mismatching using homography-based geometric alignment.

**Camera synchronization.** Even if multiple cameras are geometrically aligned, another issue arises from the fact that they can capture data at different time instances. To address this, precise time synchronization among the multi-camera setup is required, and we accomplished this by designing a micro-controller (ATmega) to serve as an external trigger for hardware-level synchronization of three different cameras. The event and two RGB cameras are connected to the micro-controller via a trigger cable, ensuring simultaneous transmission of signals. Recording software is then created using the provided C++ SDK of each camera product to control them by receiving signals from the microcontroller. Each camera synchronizes with the falling and rising edges of the trigger signals, allowing for control of the RGB camera's exposure time with synchronized signals. By using this

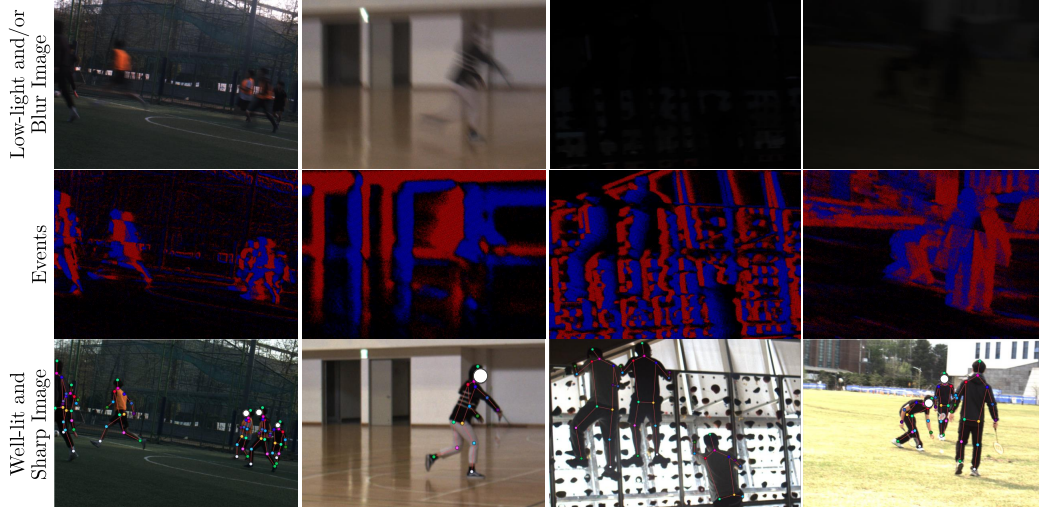

Figure 2: Visualization of sample data with pose annotations. The 1st row shows degraded RGB frames affected by motion blur and/or low-light conditions. The 2nd row displays event data captured in the same environment. The 3rd row consists of reference RGB frames for annotation, time-synchronized with the 1st row data and precisely aligned pixel-wise through a beam-splitter.

synchronization methods, we adjusted the exposure time: one camera used a short exposure time to capture a sharp image, while the other camera was set to an exposure time 16 times longer than that of the sharp image, resulting in both a blurred image and its corresponding sharp image.

## 3.2 Data Collection

To conduct our human subject study, we locally recruited participants and obtained signed consent forms and privacy agreements regarding the data distribution reviewed by the institution before the experiment. Additionally, prior to data acquisition, we reiterated the research objectives, procedures, potential risks, and data distribution to participants, informing them of their ability to withdraw from the experiment at any point. A total of 61 male and 21 female participants agreed to participate in the experiment by signing the consent form. Natural and realistic scenarios were preselected and presented to the participants, who then performed these scenarios as instructed. We acquired data corresponding to these scenarios during the participants' performance. Due to the acquisition of data in diverse lighting conditions such as indoor and outdoor environments, we did not employ fixed camera settings. Especially for low-light data, we adjusted the aperture of the camera lens and the exposure time to reduce the incoming light in the camera shutters and adjusted the gain accordingly.

**Low-light/well-lit and indoor/outdoor distributions.** The EHPT-XC aims to capture humans in various environments, including challenging low-light conditions, which can pose difficulties for pose estimation. As depicted in Fig. 3, we have split the dataset to ensure a balanced distribution of indoor/outdoor and low-light/well-lit environments between the train and test sets. Specifically, we have acquired data in such a way that low-light environments constitute a substantial portion of the overall dataset.

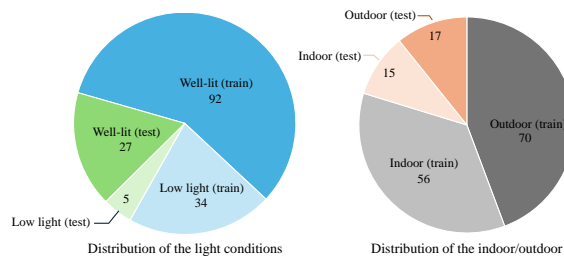

Figure 3: Scene distribution over light conditions (low-light/well-lit) and indoor/outdoor environments.

## 3.3 Data Annotation

For annotation, we utilized well-lit and sharp images as references, which were precisely geometrically and temporally aligned with the blurred and/or low-light condition images. As depicted in Fig. 2,

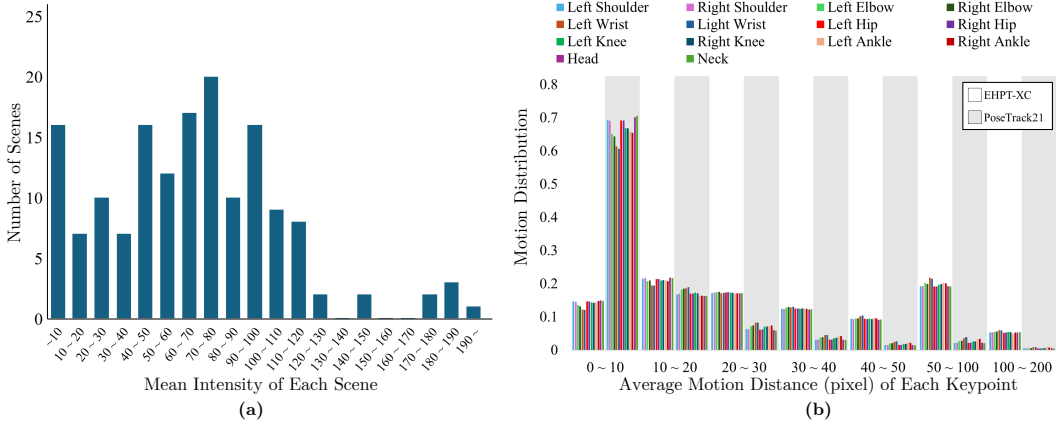

(a)

(b)

Figure 4: Statistics of EHPT-XC dataset. (a) The distribution of average channel intensity for each sequence. (b) The distribution of motion distances for each keypoint category. We compare the distribution with a recent pose estimation and tracking dataset [10].

estimating accurate key points in degraded images proved to be extremely challenging for annotators, whereas it was straightforward in the reference well-lit and sharp images. For the high-quality annotation of the EHPT-XC dataset, we engaged five annotators with ample experience in the field of computer vision. Each annotator performed annotations for different sequences, and through a cross-checking process, we enhanced the overall quality of the annotations. Following the previous labeling format [19, 37], a total of 15,800 images were labeled with 14 human skeletal keypoints.

## 3.4 Data Statistics

**Intensity distribution.** As shown in Fig. 4 (a), we calculate the average intensity of RGB frames in each scene. In fact, low-light environments have mean intensities distributed below 40, with occasional instances in indoor settings where sunlight is absent, resulting in intensities close to 40. However, upon actual inspection, these environments still provide sufficient visibility about humans. Hence, in Fig. 3, they are classified as well-lit. Low-light environments were categorized only when filters or aperture adjustments were deliberately applied to decrease the intensity. Furthermore, it can be observed that aside from the division into low-light and well-lit categories, the EHPT-XC dataset also exhibits a uniform distribution of overall intensity.

**Motion distribution.** To analyze the motion distribution of the dataset, we calculate pixel displacement for each keypoint between adjacent frames. As shown in Fig. 4 (b), we compare the motion distribution with the recent multi-human pose estimation and tracking dataset, PoseTrack21 [10]. We measure the distribution for the 14 keypoints among the 17 keypoints in PoseTrack21 for comparison.

In PoseTrack21, most of the data are distributed within the range of 0 to 10 pixels displacement, and there is minimal distribution beyond 40 pixels, which could be considered as large displacement. On the other hand, our EHPT-XC dataset encompasses a diverse range of motion distributions, including a substantial representation of large displacements, particularly those exceeding 50 pixels. This highlights EHPT-XC as a challenging dataset with a wide range of motion distributions. Moreover, it can be observed that motion is evenly distributed, encompassing various displacements for keypoints.

**Blur intensity.** The intensity of blur can be determined by the magnitude of motion

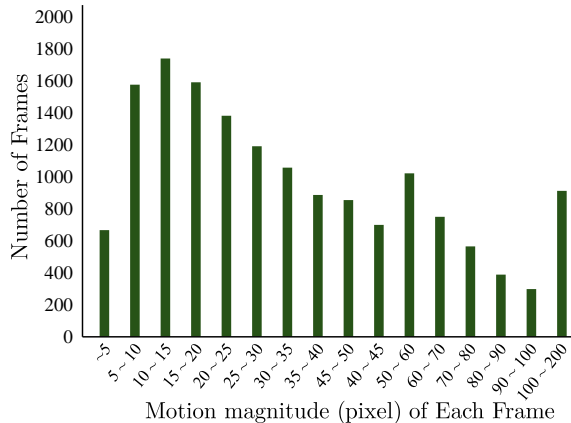

Figure 5: Motion magnitude distribution.

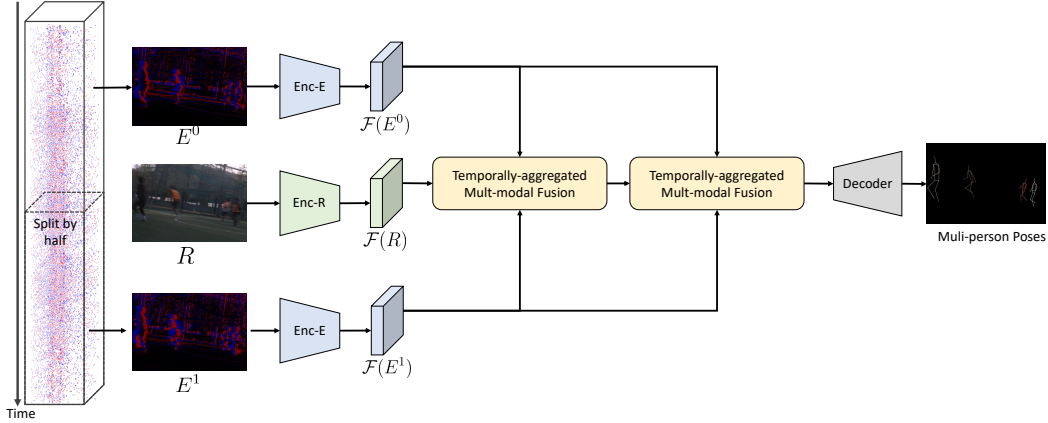

Figure 6: A baseline approach for multi-modal fusion. To achieve effective multi-modal fusion, we propose splitting the event stream into two halves centered around the midpoint in time, enabling an effective fusion with the image. Enc-E shares weights for both $E^0$ and $E^1$, and the two temporally-aggregated multi-modal fusion modules also share weights with each other.

between two adjacent sharp frames. Blur can occur not only due to the movement of the target object (*i.e.* humans), but also due to the ego motion of the camera. Therefore, to calculate blur intensity, we need to obtain the displacement of each pixel between sharp frames rather than just the keypoint displacement. To achieve this, we apply a pre-trained optical flow network [32] to sharp and well-lit reference images, calculating pixel displacement for all pixels. Figure 5 illustrates the motion magnitude of each image, calculated by averaging the displacement of all pixels. In the EHPT-XC dataset, it can be observed that the blur intensity is uniformly distributed across the entire dataset.

## 4 Baseline for multi-modal fusion

Various approaches [29, 49] have been proposed for fusing RGB and event modalities, and how the fusion between different modalities is performed significantly impacts the performance of the end task. As shown in Fig. 6, to obtain high-quality representations even from degraded inputs, we split each event into two segments at its midpoint of the exposure time and then fused them with RGB data. Given an event voxel, $E$, we split the voxel into two parts, $E^0$ and $E^1$. Then, through separate convolution-based encoders, we extract features $\mathcal{F}(E^0)$ and $\mathcal{F}(E^1)$. The two event features and one RGB feature are fed into the proposed temporally-aggregated multi-modal fusion module, where they undergo a process of being merged into a single representation. We apply this aggregation process twice consecutively, resulting in a well-fused feature even with degraded information.

**Temporally-aggregated Multi-modal Fusion.** As shown in Fig. 7, all features are concatenated and merged into a single query through a $1 \times 1$ convolution. The merged feature then undergoes a self-attention operation via an attention block. As shown in the right of Fig. 7, in the attention block, we generate query, key, and value features, $\mathbf{Q} = W_Q(\text{query})$, $\mathbf{K} = W_K(\text{key})$, $\mathbf{V} = W_V(\text{value})$, where $W_{(\cdot)}$ is $1 \times 1$ convolution with a layer normalization. Utilizing these $\mathbf{Q}$, $\mathbf{K}$, and $\mathbf{V}$, we compute the attended feature as follows:

$$\mathcal{A}(\mathbf{Q}, \mathbf{K}, \mathbf{V}) = \text{Softmax}(\frac{\mathbf{Q}^T \mathbf{K}}{\alpha}) \cdot \mathbf{V} \qquad (1)$$

where $\alpha$ is the scaling factor of the attention matrix. To reduce computational costs, we calculate the cross-covariance matrix of the attention following the method in [46]. As shown in the figure above, while self-attention is being performed, we also apply cross-attention to two event voxels, split along the temporal axis, to better fuse the modalities. All outputs are then passed through concatenation, a $1 \times 1$ convolution block, and MLP layers to generate the final feature representation.

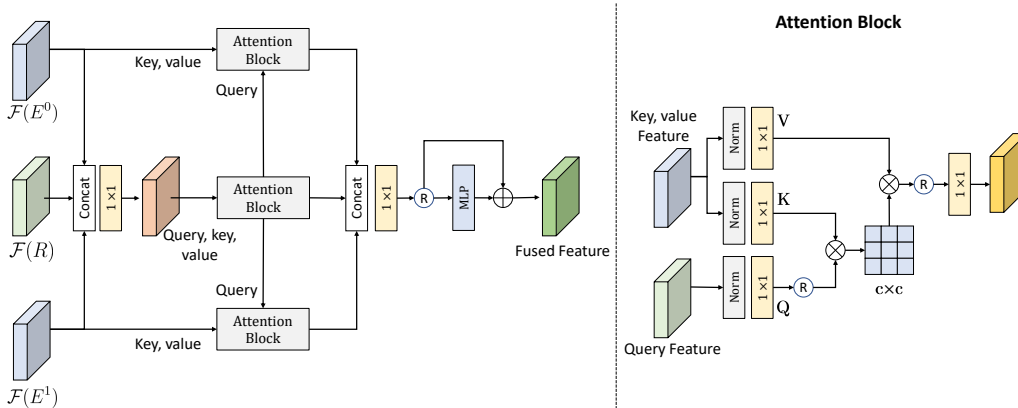

Figure 7: The proposed temporally-aggregated multi-modal fusion. First, we combine the three features and perform self-attention. Then, we apply cross-attention between each event and the combined features. Finally, we aggregate the results using an MLP layer.

Table 2: Multi-person pose estimation baselines evaluated on the EHPT-XC dataset.

| Modality | Method | mAP@0.5:0.95 | mAP@0.5 | mAP@0.75 | mAR@0.5:0.95 | mAR@0.5 | mAR@0.75 |
|---|---|---|---|---|---|---|---|
| RGB | HigherHRNet [7] | 22.7 | 31.8 | 23.3 | 67.0 | 85.8 | 70.0 |
| | DEKR [13] | 25.1 | 34.8 | 26.5 | 63.8 | 85.5 | 66.9 |
| | CID [34] | 24.0 | 33.1 | 24.5 | 65.9 | 87.7 | 67.9 |
| Event | HigherHRNet [7] | 32.1 | 37.8 | 33.6 | 83.8 | 95.5 | 86.8 |
| | DEKR [13] | 33.1 | 39.4 | 34.6 | 84.0 | 96.6 | 87.6 |
| | CID [34] | 31.1 | 38.1 | 32.6 | 85.6 | 97.0 | 88.8 |
| RGB + Event [29] | HigherHRNet [4] | 33.8 | 39.0 | 34.3 | 84.4 | 94.6 | 85.9 |
| | DEKR [9] | 36.9 | 41.0 | 37.2 | 87.7 | 95.8 | **88.9** |
| | CID [25] | 33.7 | 39.3 | 34.5 | **88.0** | **98.2** | **90.2** |
| RGB + Event (Ours) | HigherHRNet [4] | **34.3** | **39.7** | **34.8** | **86.0** | **97.0** | **87.6** |
| | DEKR [9] | **37.3** | **42.0** | **37.5** | **87.8** | **97.1** | **88.9** |
| | CID [25] | **34.5** | **40.9** | **36.0** | 84.7 | 98.0 | 88.4 |

## 5 Evaluation and Benchmarks

### 5.1 Multi-Person Pose Estimation

**Metrics.** Our evaluation strategy follows the well-established MSCOCO [20] and CrowdPose [19] metrics. We assess performance using average precision (AP) and average recall (AR). The Object Keypoint Similarity (OKS) serves a similar purpose to Intersection over Union (IoU) in the context of adopting Average Precision (AP) and Average Recall (AR) for keypoint detection. Our primary metrics are mAP and mAR, which are computed by averaging over multiple OKS values (.50:.05:.95).

**Benchmark.** We train and evaluate all methods using our images, event data, and annotations. We evaluate three main methodologies: one that utilizes only the RGB modality, another that relies solely on the event modality, and a third that integrates both RGB and event data in a multi-modality framework. To leverage the event modality, we utilized the widely used event representation, the voxel grid [56], setting the bin size to 10. For each modality, we adopt the same methods, but for the multi-modality (RGB+Event) approach, we apply three fusion methods: 1) Concatenation of input modalities. 2) Existing fusion method [29] between RGB and events. 3) Our newly proposed base fusion method (Fig. 6). We evaluate several recent state-of-the-art (SOTA) methods for multi-person pose estimation models. Specifically, we evaluate three recent bottom-up models: DEKR [13], CID [34], and HigherHRNet [7]. We remove redundant poses from the stitched annotation set by employing non-maximum suppression (NMS) on the predicted bounding boxes.

Table 2 presents the results of pose estimation, categorized by modality and method. Among methods, DEKR [13] achieves the best performance in terms of mAP metric, while HigherHRNet [7] achieves the best performance in terms of mAR metric. When examining the results of multi-modality

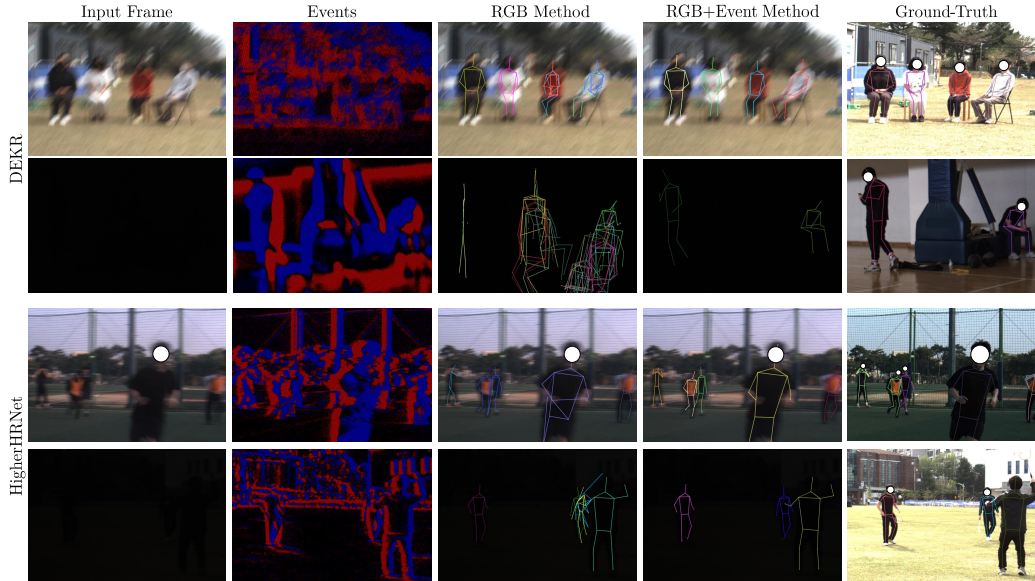

Figure 8: Qualitative results of multi-person pose estimation on the EHPT-XC dataset. The 1st and 2nd rows represent the results of DEKR [13], while the 3rd and 4th rows depict the results of HigherHRNet [7]. The ground-truth keypoints are visualized on a sharp and clean reference image used for annotations. The 2nd row presents low-light conditions with significant level of motion blur.

approaches combining RGB and event data, it's evident that performance improves across all metrics compared to using only the RGB approach. One particularly interesting observation is the significant improvement in overall performance, especially in the mAR metric, when using multi-modality approaches. This improvement suggests that the results obtained solely from RGB often fail to accurately estimate the positions of human keypoints due to motion blur and low-light conditions. In contrast, when RGB and event data are used together, even in scenarios with motion blur and low-light conditions, the positions of human joint keypoints are estimated more accurately.

Figure 8 provides a clearer representation of the performance difference between the multi-modality approaches using event data and the RGB-only method. Specifically, examining the 2nd result of the DEKR experiment, the scene presents an extremely challenging situation due to both low-light conditions and significant motion blur. Consequently, RGB-based methods tend to misinterpret the scene often leading to numerous false positive predictions where multiple humans are incorrectly identified. On the other hand, when incorporating event data, such scene misinterpretations are reduced, and keypoints are predicted accurately at the human positions. Similarly, in various challenging conditions such as severe blurring and low-light environments, multi-modal approaches demonstrate superior performance.

## 5.2 Multi-Person Pose Tracking

**Metrics.** To evaluate the pose tracking results for multiple persons, we used common evaluation metrics frequently employed in multi-object tracking, namely MOTA, IDF1, FP, IDSW, and FN. For MOTA, it provides an overall evaluation of FP, FN, and IDSW metrics, making it the most important performance indicator in multi-object tracking. Additionally, IDF1 is a metric used to assess the performance of multi-object tracking systems, focusing on the accuracy of identity matching for objects over time. It evaluates how effectively the tracker maintains the correct identities of objects throughout the tracking sequence. FP are incorrect identifications of non-existent objects, IDSW are errors where the system changes the identity of a tracked object, and FN are failures to detect existing objects. We consider MOTA as the primary metric, as done in previous benchmarks [10].

**Benchmark.** We evaluate recent state-of-the-art methodologies, ByteTrack [51], Unitrack [36], and OC-SORT [6]. These methods fall under the category of tracking-by-detection, using object detection results estimated through pose estimation methods to perform multi-object tracking. For

Table 3: Multi-person pose tracking baselines on the EHPT-XC dataset.

| Modality | Pose Estimation | Tracking | MOTA↑ | IDF1↑ | FP↓ | IDSW↓ | FN↓ |
|---|---|---|---|---|---|---|---|
| RGB | DEKR [13] | ByteTrack [51] | 33.19 | 18.73 | 328 | 316 | 4643 |
| | | UniTrack [36] | 25.60 | 8.56 | 90 | 159 | 5611 |
| | | OC-SORT [6] | 23.25 | 15.08 | 67 | 127 | 5880 |
| RGB | DEKR [13] | ByteTrack [51] | 47.37 (+14.18) | 20.46 | 461 | 405 | 3299 |
| + | | UniTrack [36] | 46.82 (+21.22) | 7.93 | 205 | 374 | 3630 |
| Event | | OC-SORT [6] | 42.34 (+19.09) | 22.72 | 193 | 207 | 4163 |

pose estimation, we adopt DEKR [13], which demonstrated satisfactory performance in both RGB and RGB+event modality approaches. To utilize both RGB and event data together, we adopted the concatenation method.

Table 3 presents the results of multi-person pose tracking, applying various tracking methods for RGB and RGB+event modalities. OC-SORT was proposed to handle occlusions by associating over long time steps. However, we believe that the relatively poor performance of OC-SORT is attributed to the significant motion displacement in the EHPT-XC dataset, where distant frames often fail to provide valuable information for current frame tracking. ByteTrack demonstrates robust operation by incorporating even low-confidence predicted bounding boxes into tracklets, thus addressing common challenges in extreme environments. When comparing the results across modalities, an interesting observation is that in extreme cases with low-light and motion blur, using only RGB data leads to relatively low false positive (FP), but significantly higher false negative (FN). When only using RGB data, objects are frequently overlooked or undetected, resulting in a higher false negative rate. On the other hand, integrating event data reduces the occurrence of missed detection, although it may introduce problems with false positives. Even considering such cases, it's evident that incorporating event data leads to a significant improvement in the overall performance of tracking, as assessed by the MOTA metric.

# 6   Conclusion and Future Work

In this paper, we establish Event-guided Human Pose Estimation and Tracking in eXtreme Conditions (EHPT-XC) dataset, which is the first multi-human pose and tracking dataset with real captured extreme motion blur and low-light conditions with the real events and frames. To leverage the benefits of event cameras, which are well-suited for such extreme environments, we customize a triplet camera system to acquire multi-modal data. Thanks to the triplet camera system, we were able to acquire clean and sharp RGB frames that are pixel-wise aligned and time-synchronized, enabling precise annotation. We benchmark recent state-of-the-art multi-person pose estimation and tracking methods on the EHPT-XC dataset, showcasing the advantages, particularly in extreme environments, of utilizing the event modality. We expect that EHPT-XC will pave the way for further exploration in understanding human actions in extreme scenarios.

**Limitation and future work.** In this work, we propose a simple multi-modal fusion baselines. However, for tasks such as human pose estimation, more sophisticated modality fusion methods may be more effective. We plan to explore these fusion methods in future work.

**Social impact.** The proposed EHPT-XC dataset aims to estimate human pose even in RGB images degraded by motion blur and/or low-light conditions, and its main applications are targeted at rehabilitation and sports. In particular, since such degraded images do not contain personal identification information, inferring poses from these images can reduce emerging privacy issues. Moreover, the event camera we adopted as part of our multi-modal approach is well-known for its privacy-preserving characteristics, making it highly suitable for this purpose. However, we are aware that it can be misused for its intended purpose (*e.g*., pedestrian surveillance at night), and in case of misuse, we reserve our right to withdraw permission for users to use the dataset at any point.

# Acknowledgement

This work was supported by the National Research Foundation of Korea (NRF) grant funded by the Korea government (MSIT) (NRF2022R1A2B5B03002636).

# References

[1] S. An and U. Y. Ogras. Mars: mmwave-based assistive rehabilitation system for smart healthcare. *ACM Transactions on Embedded Computing Systems (TECS)*, 20(5s):1–22, 2021.

[2] S. An, Y. Li, and U. Ogras. mri: Multi-modal 3d human pose estimation dataset using mmwave, rgb-d, and inertial sensors. *Advances in Neural Information Processing Systems*, 35:27414–27426, 2022.

[3] M. Andriluka, L. Pishchulin, P. Gehler, and B. Schiele. 2d human pose estimation: New benchmark and state of the art analysis. In *Proceedings of the IEEE Conference on computer Vision and Pattern Recognition*, pages 3686–3693, 2014.

[4] Z. Cai, D. Ren, A. Zeng, Z. Lin, T. Yu, W. Wang, X. Fan, Y. Gao, Y. Yu, L. Pan, et al. Humman: Multi-modal 4d human dataset for versatile sensing and modeling. In *European Conference on Computer Vision*, pages 557–577. Springer, 2022.

[5] E. Calabrese, G. Taverni, C. Awai Easthope, S. Skriabine, F. Corradi, L. Longinotti, K. Eng, and T. Delbruck. Dhp19: Dynamic vision sensor 3d human pose dataset. In *Proceedings of the IEEE/CVF conference on computer vision and pattern recognition workshops*, pages 0–0, 2019.

[6] J. Cao, J. Pang, X. Weng, R. Khirodkar, and K. Kitani. Observation-centric sort: Rethinking sort for robust multi-object tracking. In *Proceedings of the IEEE/CVF conference on computer vision and pattern recognition*, pages 9686–9696, 2023.

[7] B. Cheng, B. Xiao, J. Wang, H. Shi, T. S. Huang, and L. Zhang. Higherhrnet: Scale-aware representation learning for bottom-up human pose estimation. In *Proceedings of the IEEE/CVF conference on computer vision and pattern recognition*, pages 5386–5395, 2020.

[8] H. Cui, S. Zhong, J. Wu, Z. Shen, N. Dahnoun, and Y. Zhao. Milipoint: A point cloud dataset for mmwave radar. In *Advances in Neural Information Processing Systems*, volume 36, pages 62713–62726, 2023.

[9] P. Dendorfer, H. Rezatofighi, A. Milan, J. Shi, D. Cremers, I. Reid, S. Roth, K. Schindler, and L. Leal-Taixé. Mot20: A benchmark for multi object tracking in crowded scenes. *arXiv preprint arXiv:2003.09003*, 2020.

[10] A. Doering, D. Chen, S. Zhang, B. Schiele, and J. Gall. Posetrack21: A dataset for person search, multi-object tracking and multi-person pose tracking. In *Proceedings of the IEEE/CVF Conference on Computer Vision and Pattern Recognition*, pages 20963–20972, 2022.

[11] M. Fabbri, G. Brasó, G. Maugeri, O. Cetintas, R. Gasparini, A. Ošep, S. Calderara, L. Leal-Taixé, and R. Cucchiara. Motsynth: How can synthetic data help pedestrian detection and tracking? In *Proceedings of the IEEE/CVF International Conference on Computer Vision*, pages 10849–10859, 2021.

[12] G. Gallego, T. Delbrück, G. Orchard, C. Bartolozzi, B. Taba, A. Censi, S. Leutenegger, A. J. Davison, J. Conradt, K. Daniilidis, et al. Event-based vision: A survey. *IEEE transactions on pattern analysis and machine intelligence*, 44(1):154–180, 2020.

[13] Z. Geng, K. Sun, B. Xiao, Z. Zhang, and J. Wang. Bottom-up human pose estimation via disentangled keypoint regression. In *Proceedings of the IEEE/CVF conference on computer vision and pattern recognition*, pages 14676–14686, 2021.

[14] W. Jiang, H. Xue, C. Miao, S. Wang, S. Lin, C. Tian, S. Murali, H. Hu, Z. Sun, and L. Su. Towards 3d human pose construction using wifi. In *Proceedings of the 26th Annual International Conference on Mobile Computing and Networking*, pages 1–14, 2020.

[15] T. Kim, H. Cho, and K.-J. Yoon. Cross-modal temporal alignment for event-guided video deblurring. *arXiv preprint arXiv:2408.14930*, 2024.

[16] T. Kim, H. Cho, and K.-J. Yoon. Frequency-aware event-based video deblurring for real-world motion blur. In *Proceedings of the IEEE/CVF Conference on Computer Vision and Pattern Recognition*, pages 24966–24976, 2024.

[17] S. Lee, J. Rim, B. Jeong, G. Kim, B. Woo, H. Lee, S. Cho, and S. Kwak. Human pose estimation in extremely low-light conditions. In *Proceedings of the IEEE/CVF Conference on Computer Vision and Pattern Recognition*, pages 704–714, 2023.

[18] S.-P. Lee, N. P. Kini, W.-H. Peng, C.-W. Ma, and J.-N. Hwang. Hupr: A benchmark for human pose estimation using millimeter wave radar. In *Proceedings of the IEEE/CVF Winter Conference on Applications of Computer Vision*, pages 5715–5724, 2023.

[19] J. Li, C. Wang, H. Zhu, Y. Mao, H.-S. Fang, and C. Lu. Crowdpose: Efficient crowded scenes pose estimation and a new benchmark. In *Proceedings of the IEEE/CVF conference on computer vision and pattern recognition*, pages 10863–10872, 2019.

[20] T.-Y. Lin, M. Maire, S. Belongie, J. Hays, P. Perona, D. Ramanan, P. Dollár, and C. L. Zitnick. Microsoft coco: Common objects in context. In *Computer Vision–ECCV 2014: 13th European Conference, Zurich, Switzerland, September 6-12, 2014, Proceedings, Part V 13*, pages 740–755. Springer, 2014.

[21] J. Liu, D. Xu, W. Yang, M. Fan, and H. Huang. Benchmarking low-light image enhancement and beyond. *International Journal of Computer Vision*, 129:1153–1184, 2021.

[22] J. Liu, X. Fan, Z. Huang, G. Wu, R. Liu, W. Zhong, and Z. Luo. Target-aware dual adversarial learning and a multi-scenario multi-modality benchmark to fuse infrared and visible for object detection. In *Proceedings of the IEEE/CVF conference on computer vision and pattern recognition*, pages 5802–5811, 2022.

[23] Y. P. Loh and C. S. Chan. Getting to know low-light images with the exclusively dark dataset. *Computer Vision and Image Understanding*, 178:30–42, 2019.

[24] J. S. Lumentut and I. K. Park. Human and scene motion deblurring using pseudo-blur synthesizer. *IEEE Access*, 9:146366–146377, 2021.

[25] J. S. Lumentut, J. Santoso, and I. K. Park. Human motion deblurring using localized body prior. In *Proceedings of the Asian Conference on Computer Vision*, 2020.

[26] Y. Ren, Z. Wang, Y. Wang, S. Tan, Y. Chen, and J. Yang. Gopose: 3d human pose estimation using wifi. *Proceedings of the ACM on Interactive, Mobile, Wearable and Ubiquitous Technologies*, 6(2):1–25, 2022.

[27] J. Rim, H. Lee, J. Won, and S. Cho. Real-world blur dataset for learning and benchmarking deblurring algorithms. In *Computer Vision–ECCV 2020: 16th European Conference, Glasgow, UK, August 23–28, 2020, Proceedings, Part XXV 16*, pages 184–201. Springer, 2020.

[28] A. Shahroudy, J. Liu, T.-T. Ng, and G. Wang. Ntu rgb+ d: A large scale dataset for 3d human activity analysis. In *Proceedings of the IEEE conference on computer vision and pattern recognition*, pages 1010–1019, 2016.

[29] L. Sun, C. Sakaridis, J. Liang, Q. Jiang, K. Yang, P. Sun, Y. Ye, K. Wang, and L. V. Gool. Event-based fusion for motion deblurring with cross-modal attention. In *European conference on computer vision*, pages 412–428. Springer, 2022.

[30] P. Sun, H. Kretzschmar, X. Dotiwalla, A. Chouard, V. Patnaik, P. Tsui, J. Guo, Y. Zhou, Y. Chai, B. Caine, et al. Scalability in perception for autonomous driving: Waymo open dataset. In *Proceedings of the IEEE/CVF conference on computer vision and pattern recognition*, pages 2446–2454, 2020.

[31] C. Tang, X. Wang, J. Huang, B. Jiang, L. Zhu, J. Zhang, Y. Wang, and Y. Tian. Revisiting color-event based tracking: A unified network, dataset, and metric. *arXiv preprint arXiv:2211.11010*, 2022.

[32] Z. Teed and J. Deng. Raft: Recurrent all-pairs field transforms for optical flow. In *Computer Vision–ECCV 2020: 16th European Conference, Glasgow, UK, August 23–28, 2020, Proceedings, Part II 16*, pages 402–419. Springer, 2020.

[33] E. Vendrow, D. T. Le, J. Cai, and H. Rezatofighi. Jrdb-pose: A large-scale dataset for multi-person pose estimation and tracking. In *Proceedings of the IEEE/CVF Conference on Computer Vision and Pattern Recognition*, pages 4811–4820, 2023.

[34] D. Wang and S. Zhang. Contextual instance decoupling for robust multi-person pose estimation. In *Proceedings of the IEEE/CVF Conference on Computer Vision and Pattern Recognition*, pages 11060–11068, 2022.

[35] X. Wang, J. Li, L. Zhu, Z. Zhang, Z. Chen, X. Li, Y. Wang, Y. Tian, and F. Wu. Visevent: Reliable object tracking via collaboration of frame and event flows. *IEEE Transactions on Cybernetics*, 2023.

[36] Z. Wang, H. Zhao, Y.-L. Li, S. Wang, P. Torr, and L. Bertinetto. Do different tracking tasks require different appearance models? *Advances in neural information processing systems*, 34:726–738, 2021.

[37] J. Wu, H. Zheng, B. Zhao, Y. Li, B. Yan, R. Liang, W. Wang, S. Zhou, G. Lin, Y. Fu, et al. Ai challenger: A large-scale dataset for going deeper in image understanding. *arXiv preprint arXiv:1711.06475*, 2017.

[38] C. T. L. Z. B. J. Y. T. J. T. Xiao Wang, Shiao Wang. Event stream-based visual object tracking: A high-resolution benchmark dataset and a novel baseline. *arXiv:2309.14611*, 2023. URL `https://arxiv.org/abs/2309.14611`.

[39] L. Xu, W. Xu, V. Golyanik, M. Habermann, L. Fang, and C. Theobalt. Eventcap: Monocular 3d capture of high-speed human motions using an event camera. In *Proceedings of the IEEE/CVF Conference on Computer Vision and Pattern Recognition*, pages 4968–4978, 2020.

[40] X. Xu, S. Wang, Z. Wang, X. Zhang, and R. Hu. Exploring image enhancement for salient object detection in low light images. *ACM transactions on multimedia computing, communications, and applications (TOMM)*, 17(1s):1–19, 2021.

[41] X. Xu, X. Wei, Y. Xu, Z. Zhang, K. Gong, H. Li, and L. Xiao. Infpose: Real-time infrared multi-human pose estimation for edge devices based on encoder-decoder cnn architecture. *IEEE Robotics and Automation Letters*, 2023.

[42] M. Yan, Y. Zhang, S. Cai, S. Fan, X. Lin, Y. Dai, S. Shen, C. Wen, L. Xu, Y. Ma, et al. Reli11d: A comprehensive multimodal human motion dataset and method. In *Proceedings of the IEEE/CVF Conference on Computer Vision and Pattern Recognition*, pages 2250–2262, 2024.

[43] J. Yang, H. Huang, Y. Zhou, X. Chen, Y. Xu, S. Yuan, H. Zou, C. X. Lu, and L. Xie. Mm-fi: Multi-modal non-intrusive 4d human dataset for versatile wireless sensing. *Advances in Neural Information Processing Systems*, 36, 2024.

[44] M. Yang, S.-C. Liu, and T. Delbruck. A dynamic vision sensor with 1% temporal contrast sensitivity and in-pixel asynchronous delta modulator for event encoding. *IEEE Journal of Solid-State Circuits*, 50(9): 2149–2160, 2015.

[45] W. Yang, Y. Yuan, W. Ren, J. Liu, W. J. Scheirer, Z. Wang, T. Zhang, Q. Zhong, D. Xie, S. Pu, et al. Advancing image understanding in poor visibility environments: A collective benchmark study. *IEEE Transactions on Image Processing*, 29:5737–5752, 2020.

[46] S. W. Zamir, A. Arora, S. Khan, M. Hayat, F. S. Khan, and M.-H. Yang. Restormer: Efficient transformer for high-resolution image restoration. In *CVPR*, 2022.

[47] J. Zhang, X. Yang, Y. Fu, X. Wei, B. Yin, and B. Dong. Object tracking by jointly exploiting frame and event domain. In *Proceedings of the IEEE/CVF International Conference on Computer Vision*, pages 13043–13052, 2021.

[48] J. Zhang, B. Dong, H. Zhang, J. Ding, F. Heide, B. Yin, and X. Yang. Spiking transformers for event-based single object tracking. In *Proceedings of the IEEE/CVF conference on Computer Vision and Pattern Recognition*, pages 8801–8810, 2022.

[49] J. Zhang, Y. Wang, W. Liu, M. Li, J. Bai, B. Yin, and X. Yang. Frame-event alignment and fusion network for high frame rate tracking. In *Proceedings of the IEEE/CVF Conference on Computer Vision and Pattern Recognition*, pages 9781–9790, 2023.

[50] W. Zhang, M. Zhu, and K. G. Derpanis. From actemes to action: A strongly-supervised representation for detailed action understanding. In *Proceedings of the IEEE international conference on computer vision*, pages 2248–2255, 2013.

[51] Y. Zhang, P. Sun, Y. Jiang, D. Yu, F. Weng, Z. Yuan, P. Luo, W. Liu, and X. Wang. Bytetrack: Multi-object tracking by associating every detection box. In *European conference on computer vision*, pages 1–21. Springer, 2022.

[52] M. Zhao, T. Li, M. Abu Alsheikh, Y. Tian, H. Zhao, A. Torralba, and D. Katabi. Through-wall human pose estimation using radio signals. In *Proceedings of the IEEE conference on computer vision and pattern recognition*, pages 7356–7365, 2018.

[53] Y. Zhao, D. Rozumnyi, J. Song, O. Hilliges, M. Pollefeys, and M. R. Oswald. Human from blur: Human pose tracking from blurry images. In *Proceedings of the IEEE/CVF International Conference on Computer Vision*, pages 14905–14915, 2023.

[54] Z. Zhong, Y. Gao, Y. Zheng, and B. Zheng. Efficient spatio-temporal recurrent neural network for video deblurring. In *Computer Vision–ECCV 2020: 16th European Conference, Glasgow, UK, August 23–28, 2020, Proceedings, Part VI 16*, pages 191–207. Springer, 2020.

[55] Z. Zhong, M. Cao, X. Ji, Y. Zheng, and I. Sato. Blur interpolation transformer for real-world motion from blur. In *Proceedings of the IEEE/CVF Conference on Computer Vision and Pattern Recognition*, pages 5713–5723, 2023.

[56] A. Z. Zhu, L. Yuan, K. Chaney, and K. Daniilidis. Unsupervised event-based learning of optical flow, depth, and egomotion. In *Proceedings of the IEEE/CVF Conference on Computer Vision and Pattern Recognition*, pages 989–997, 2019.

[57] S. Zou, C. Guo, X. Zuo, S. Wang, P. Wang, X. Hu, S. Chen, M. Gong, and L. Cheng. Eventhpe: Event-based 3d human pose and shape estimation. In *Proceedings of the IEEE/CVF International Conference on Computer Vision*, pages 10996–11005, 2021.

